# An Analysis of Convex Relaxations for MAP Estimation

**M. Pawan Kumar**
**Dept. of Computing**
**Oxford Brookes University**
**pkmudigonda@brookes.ac.uk**

**V. Kolmogorov**
**Computer Science**
**University College London**
**vnk@adastral.ucl.ac.uk**

**P.H.S. Torr**
**Dept. of Computing**
**Oxford Brookes University**
**philiptorr@brookes.ac.uk**

## Abstract

The problem of obtaining the maximum a posteriori estimate of a general discrete random field (i.e. a random field defined using a finite and discrete set of labels) is known to be NP-hard. However, due to its central importance in many applications, several approximate algorithms have been proposed in the literature. In this paper, we present an analysis of three such algorithms based on convex relaxations: (i) LP-S: the linear programming (LP) relaxation proposed by Schlesinger [20] for a special case and independently in [4, 12, 23] for the general case; (ii) QP-RL: the quadratic programming (QP) relaxation by Ravikumar and Lafferty [18]; and (iii) SOCP-MS: the second order cone programming (SOCP) relaxation first proposed by Muramatsu and Suzuki [16] for two label problems and later extended in [14] for a general label set.

We show that the SOCP-MS and the QP-RL relaxations are equivalent. Furthermore, we prove that despite the flexibility in the form of the constraints/objective function offered by QP and SOCP, the LP-S relaxation *strictly dominates* (i.e. provides a better approximation than) QP-RL and SOCP-MS. We generalize these results by defining a large class of SOCP (and equivalent QP) relaxations which is dominated by the LP-S relaxation. Based on these results we propose some novel SOCP relaxations which strictly dominate the previous approaches.

## 1 Introduction

Discrete random fields are a powerful tool to obtain a probabilistic formulation for various applications in Computer Vision and related areas [3]. Hence, developing accurate and efficient algorithms for performing inference on a given discrete random field is of fundamental importance. In this work, we will focus on the problem of maximum a posteriori (MAP) estimation. MAP estimation is a key step in obtaining the solutions to many applications such as stereo, image stitching and segmentation [21]. Furthermore, it is closely related to many important Combinatorial Optimization problems such as MAXCUT [6], multi-way cut [5], metric labelling [3, 11] and 0-extension [3, 9].

Given data $\mathbf{D}$, a discrete random field models the distribution (i.e. either the joint or the conditional probability) of a labelling for a set of random variables. Each of these variables $\mathbf{v} = \{v_0, v_1, \cdots, v_{n-1}\}$ can take a label from a discrete set $\mathbf{l} = \{l_0, l_1, \cdots, l_{h-1}\}$. A particular labelling of variables $\mathbf{v}$ is specified by a function $f$ whose domain corresponds to the indices of the random variables and whose range is the index of the label set, i.e. $f : \{0, 1, \cdots, n-1\} \to \{0, 1, \cdots, h-1\}$. In other words, random variable $v_a$ takes label $l_{f(a)}$. For convenience, we assume the model to be a conditional random field (CRF) while noting that all the results of this paper also apply to Markov random fields (MRF).

A CRF specifies a neighbourhood relationship $\mathcal{E}$ between the random variables, i.e. $(a, b) \in \mathcal{E}$ if, and only if, $v_a$ and $v_b$ are neighbouring random variables. Within this framework, the conditional probability of a labelling $f$ given data $\mathbf{D}$ is specified as $\Pr(f|\mathbf{D}, \boldsymbol{\theta}) = \frac{1}{Z(\boldsymbol{\theta})} \exp(-Q(f; \mathbf{D}, \boldsymbol{\theta}))$. Here $\boldsymbol{\theta}$ represents the parameters of the CRF and $Z(\boldsymbol{\theta})$ is a normalization constant which ensures that the probability sums to one (also known as the partition function). The energy $Q(f; \mathbf{D}, \boldsymbol{\theta})$ is given by $Q(f; \mathbf{D}, \boldsymbol{\theta}) = \sum_{v_a \in \mathbf{v}} \theta^1_{a;f(a)} + \sum_{(a,b) \in \mathcal{E}} \theta^2_{ab;f(a)f(b)}$. The term $\theta^1_{a;f(a)}$ is called a unary potential since its value depends on the labelling of one random variable at a time. Similarly, $\theta^2_{ab;f(a)f(b)}$ is called a pairwise potential as it depends on a pair of random variables. For simplicity, we assume

that $\theta^2_{ab;f(a)f(b)} = w(a,b)d(f(a),f(b))$ where $w(a,b)$ is the weight that indicates the strength of the pairwise relationship between variables $v_a$ and $v_b$, with $w(a,b) = 0$ if $(a,b) \notin \mathcal{E}$, and $d(\cdot,\cdot)$ is a distance function on the labels. As will be seen later, this formulation of the pairwise potentials would allow us to concisely describe our results.

The problem of MAP estimation is well known to be NP-hard in general. Since it plays a central role in several applications, many approximate algorithms have been proposed in the literature. In this work, we analyze three such algorithms which are based on convex relaxations. Specifically, we consider: (i) LP-S, the linear programming (LP) relaxation of [4, 12, 20, 23]; (ii) QP-RL, the quadratic programming (QP) relaxation of [18]; and (iii) SOCP-MS, the second order cone programming (SOCP) relaxation of [14, 16]. In order to provide an outline of these relaxations, we formulate the problem of MAP estimation as an Integer Program (IP).

## 1.1 Integer Programming Formulation

We define a binary variable vector $\mathbf{x}$ of length $nh$. We denote the element of $\mathbf{x}$ at index $a \cdot h + i$ as $x_{a;i}$ where $v_a \in \mathbf{v}$ and $l_i \in \mathbf{l}$. These elements $x_{a;i}$ specify a labelling $f$ such that $x_{a;i} = 1$ if $f(a) = i$ and $x_{a;i} = -1$ otherwise. We say that the variable $x_{a;i}$ *belongs to* variable $v_a$ since it defines which label $v_a$ does (or does not) take. Let $\mathbf{X} = \mathbf{x}\mathbf{x}^\top$. We refer to the $(a \cdot h + i, b \cdot h + j)^{th}$ element of the matrix $\mathbf{X}$ as $X_{ab;ij}$ where $v_a, v_b \in \mathbf{v}$ and $l_i, l_j \in \mathbf{l}$. Clearly, the following IP finds the labelling with the minimum energy, i.e. it is equivalent to the MAP estimation problem:

$$\text{IP:} \quad \mathbf{x}^* = \arg\min_\mathbf{x} \sum_{v_a, l_i} \theta^1_{a;i} \frac{(1 + x_{a;i})}{2} + \sum_{(a,b) \in \mathcal{E}, l_i, l_j} \theta^2_{ab;ij} \frac{(1 + x_{a;i} + x_{b;j} + X_{ab;ij})}{4}$$

$$\text{s.t.} \quad \mathbf{x} \in \{-1, 1\}^{nh}, \tag{1}$$

$$\sum_{l_i \in \mathbf{l}} x_{a;i} = 2 - h, \tag{2}$$

$$\mathbf{X} = \mathbf{x}\mathbf{x}^\top. \tag{3}$$

Constraints (1) and (3) specify that the variables $\mathbf{x}$ and $\mathbf{X}$ are binary such that $X_{ab;ij} = x_{a;i}x_{b;j}$. We will refer to them as the *integer constraints*. Constraint (2), which specifies that each variable should be assigned only one label, is known as the *uniqueness constraint*. Note that one uniqueness constraint is specified for each variable $v_a$. Solving the above IP is in general NP-hard. It is therefore common practice to obtain an approximate solution using convex relaxations. We describe four such convex relaxations below.

## 1.2 Linear Programming Relaxation

The LP relaxation (proposed by Schlesinger [20] for a special case and independently in [4, 12, 23] for the general case), which we call LP-S, is given as follows:

$$\text{LP-S:} \quad \mathbf{x}^* = \arg\min_\mathbf{x} \sum_{v_a, l_i} \theta^1_{a;i} \frac{(1 + x_{a;i})}{2} + \sum_{(a,b) \in \mathcal{E}, l_i, l_j} \theta^2_{ab;ij} \frac{(1 + x_{a;i} + x_{b;j} + X_{ab;ij})}{4}$$

$$\text{s.t.} \quad \mathbf{x} \in [-1, 1]^{nh}, \mathbf{X} \in [-1, 1]^{nh \times nh}, \tag{4}$$

$$\sum_{l_i \in \mathbf{l}} x_{a;i} = 2 - h, \tag{5}$$

$$\sum_{l_j \in \mathbf{l}} X_{ab;ij} = (2 - h)x_{a;i}, \tag{6}$$

$$X_{ab;ij} = X_{ba;ji}, \tag{7}$$

$$1 + x_{a;i} + x_{b;j} + X_{ab;ij} \geq 0. \tag{8}$$

In the LP-S relaxation only those elements $X_{ab;ij}$ of $\mathbf{X}$ are used for which $(a,b) \in \mathcal{E}$ and $l_i, l_j \in \mathbf{l}$. Unlike the IP, the feasibility region of the above problem is relaxed such that the variables $x_{a;i}$ and $X_{ab;ij}$ lie in the interval $[-1, 1]$. Further, the constraint (3) is replaced by equation (6) which is called the *marginalization constraint* [23]. One marginalization constraint is specified for each $(a,b) \in \mathcal{E}$ and $l_i \in \mathbf{l}$. Constraint (7) specifies that $\mathbf{X}$ is symmetric. Constraint (8) ensures that $\theta^2_{ab;ij}$ is multiplied by a number between 0 and 1 in the objective function. These constraints (7) and (8) are defined for all $(a,b) \in \mathcal{E}$ and $l_i, l_j \in \mathbf{l}$. Note that the above constraints are not exhaustive, i.e. it is possible to specify other constraints for the problem of MAP estimation (as will be seen in the different relaxations described in the subsequent sections).

## 1.3 Quadratic Programming Relaxation

We now describe the QP relaxation for the MAP estimation IP which was proposed by Ravikumar and Lafferty [18]. To this end, it would be convenient to reformulate the objective function of the IP using a vector of unary potentials of length $nh$ (denoted by $\hat{\boldsymbol{\theta}}^1$) and a matrix of pairwise potentials

of size $nh \times nh$ (denoted by $\hat{\boldsymbol{\theta}}^2$). The element of the unary potential vector at index $(a \cdot h + i)$ is defined as $\hat{\theta}^1_{a;i} = \theta^1_{a;i} - \sum_{v_c \in \mathbf{v}} \sum_{l_k \in \mathbf{l}} |\theta^2_{ac;ik}|$, where $v_a \in \mathbf{v}$ and $l_i \in \mathbf{l}$. The $(a \cdot h + i, b \cdot h + j)^{th}$ element of the pairwise potential matrix $\hat{\boldsymbol{\theta}}^2$ is defined such that

$$\hat{\theta}^2_{ab;ij} = \begin{cases} \sum_{v_c \in \mathbf{v}} \sum_{l_k \in \mathbf{l}} |\theta^2_{ac;ik}|, & \text{if} \quad a = b, i = j, \\ \theta^2_{ab;ij} & \text{otherwise,} \end{cases} \tag{9}$$

where $v_a, v_b \in \mathbf{v}$ and $l_i, l_j \in \mathbf{l}$. In other words, the potentials are modified by defining a pairwise potential $\hat{\theta}^2_{aa;ii}$ and subtracting the value of that potential from the corresponding unary potential $\theta^1_{a;i}$. The advantage of this reformulation is that the matrix $\hat{\boldsymbol{\theta}}^2$ is guaranteed to be positive semidefinite, i.e. $\hat{\boldsymbol{\theta}}^2 \succeq 0$. Using the fact that for $x_{a;i} \in \{-1, 1\}$, $\left(\frac{1+x_{a;i}}{2}\right)^2 = \frac{1+x_{a;i}}{2}$, it can be shown that the following is equivalent to the MAP estimation problem [18]:

$$\text{QP-RL:} \qquad \mathbf{x}^* = \arg\min_{\mathbf{x}} \left(\frac{\mathbf{1}+\mathbf{x}}{2}\right)^\top \hat{\boldsymbol{\theta}}^1 + \left(\frac{\mathbf{1}+\mathbf{x}}{2}\right)^\top \hat{\boldsymbol{\theta}}^2 \left(\frac{\mathbf{1}+\mathbf{x}}{2}\right), \tag{10}$$

$$\text{s.t.} \qquad \sum_{l_i \in \mathbf{l}} x_{a;i} = 2 - h, \forall v_a \in \mathbf{v}, \tag{11}$$

$$\mathbf{x} \in \{-1, 1\}^{nh}, \tag{12}$$

where $\mathbf{1}$ is a vector of appropriate dimensions whose elements are all equal to 1. By relaxing the feasibility region of the above problem to $\mathbf{x} \in [-1, 1]^{nh}$, the resulting QP can be solved in polynomial time since $\hat{\boldsymbol{\theta}}^2 \succeq 0$ (i.e. the relaxation of the QP (10)-(12) is convex). We call the above relaxation QP-RL. Note that in [18], the QP-RL relaxation was described using the variable $\mathbf{y} = \frac{\mathbf{1}+\mathbf{x}}{2}$. However, the above formulation can easily be shown to be equivalent to the one presented in [18].

## 1.4 Semidefinite Programming Relaxation

The SDP relaxation of the MAP estimation problem replaces the non-convex constraint $\mathbf{X} = \mathbf{x}\mathbf{x}^\top$ by the convex semidefinite constraint $\mathbf{X} - \mathbf{x}\mathbf{x}^\top \succeq 0$ [6, 15] which can be expressed as

$$\begin{pmatrix} 1 & \mathbf{x}^\top \\ \mathbf{x} & \mathbf{X} \end{pmatrix} \succeq 0, \tag{13}$$

using Schur's complement [2]. Further, like LP-S, it relaxes the integer constraints by allowing the variables $x_{a;i}$ and $X_{ab;ij}$ to lie in the interval $[-1, 1]$ with $X_{aa;ii} = 1$ for all $v_a \in \mathbf{v}, l_i \in \mathbf{l}$. The SDP relaxation is a well-studied approach which provides accurate solutions for the MAP estimation problem (e.g. see [25]). However, due to its computational inefficiency, it is not practically useful for large scale problems with $nh > 1000$. See however [17, 19, 22].

## 1.5 Second Order Cone Programming Relaxation

We now describe the SOCP relaxation that was proposed by Muramatsu and Suzuki [16] for the MAXCUT problem (i.e. MAP estimation with $h = 2$) and later extended for a general label set [14]. This relaxation, which we call SOCP-MS, is based on the technique of Kim and Kojima [10] who observed that the SDP constraint can be further relaxed to second order cone (SOC) constraints. For this purpose, it employs a set of matrices $\mathcal{S} = \{\mathbf{C}^k | \mathbf{C}^k = \mathbf{U}^k (\mathbf{U}^k)^\top \succeq 0, k = 1, 2, \ldots, n_C\}$. Using the fact that the Frobenius dot product of two semidefinite matrices is non-negative, we get

$$\Rightarrow \|(\mathbf{U}^k)^\top \mathbf{x}\|^2 \leq \mathbf{C}^k \bullet \mathbf{X}, k = 1, \cdots, n_C. \tag{14}$$

Each of the above SOC constraints may involve some or all variables $x_{a;i}$ and $X_{ab;ij}$. For example, if $C^k_{ab;ij} = 0$, then the $k^{th}$ SOC constraint will not involve $X_{ab;ij}$ (since its coefficient will be 0).

In order to describe the SOCP-MS relaxation, we consider a pair of neighbouring variables $v_a$ and $v_b$, i.e. $(a, b) \in \mathcal{E}$, and a pair of labels $l_i$ and $l_j$. These two pairs define the following variables: $x_{a;i}, x_{b;j}, X_{aa;ii} = X_{bb;jj} = 1$ and $X_{ab;ij} = X_{ba;ji}$ (since $\mathbf{X}$ is symmetric). For each such pair of variables and labels, the SOCP-MS relaxation specifies two SOC constraints which involve only the above variables [14, 16]. In order to specify the exact form of these SOC constraints, we need the following definitions.

Using the variables $v_a$ and $v_b$ (where $(a, b) \in \mathcal{E}$) and labels $l_i$ and $l_j$, we define the submatrices $\mathbf{x}^{(a,b,i,j)}$ and $\mathbf{X}^{(a,b,i,j)}$ of $\mathbf{x}$ and $\mathbf{X}$ respectively as:

$$\mathbf{x}^{(a,b,i,j)} = \begin{pmatrix} x_{a;i} \\ x_{b;j} \end{pmatrix}, \mathbf{X}^{(a,b,i,j)} = \begin{pmatrix} X_{aa;ii} & X_{ab;ij} \\ X_{ba;ji} & X_{bb;jj} \end{pmatrix}. \tag{15}$$

The SOCP-MS relaxation specifies SOC constraints of the form (14) for all pairs of neighbouring variables $(a, b) \in \mathcal{E}$ and labels $l_i, l_j \in \mathbf{l}$. To this end, it uses the following two matrices: $\mathbf{C}_{MS}^1 = \begin{pmatrix} 1 & 1 \\ 1 & 1 \end{pmatrix}, \mathbf{C}_{MS}^2 = \begin{pmatrix} 1 & -1 \\ -1 & 1 \end{pmatrix}$. Hence, in the SOCP-MS formulation, the MAP estimation IP is relaxed to

$$\text{SOCP-MS:} \quad \mathbf{x}^* = \arg\min_{\mathbf{x}} \sum_{v_a, l_i} \theta_{a;i}^1 \frac{(1+x_{a;i})}{2} + \sum_{(a,b) \in \mathcal{E}, l_i, l_j} \theta_{ab;ij}^2 \frac{(1+x_{a;i}+x_{b;j}+X_{ab;ij})}{4}$$

$$\text{s.t.} \qquad \mathbf{x} \in [-1, 1]^{nh}, \mathbf{X} \in [-1, 1]^{nh \times nh}, \tag{16}$$

$$\sum_{l_i \in \mathbf{l}} x_{a;i} = 2 - h, \tag{17}$$

$$(x_{a;i} - x_{b;j})^2 \leq 2 - 2X_{ab;ij}, \tag{18}$$

$$(x_{a;i} + x_{b;j})^2 \leq 2 + 2X_{ab;ij}, \tag{19}$$

$$X_{ab;ij} = X_{ba;ji}. \tag{20}$$

We refer the reader to [14, 16] for details.

## 2  Comparing Relaxations

In order to compare the relaxations described above, we require the following definitions. We say that a relaxation A *dominates* the relaxation B (alternatively, B is dominated by A) if and only if

$$\min_{(\mathbf{x}, \mathbf{X}) \in \mathcal{F}(\text{A})} e(\mathbf{x}, \mathbf{X}; \boldsymbol{\theta}) \geq \min_{(\mathbf{x}, \mathbf{X}) \in \mathcal{F}(\text{B})} e(\mathbf{x}, \mathbf{X}; \boldsymbol{\theta}), \forall \boldsymbol{\theta}, \tag{21}$$

where $\mathcal{F}(\text{A})$ and $\mathcal{F}(\text{B})$ are the feasibility regions of the relaxations A and B respectively. The term $e(\mathbf{x}, \mathbf{X}; \boldsymbol{\theta})$ denotes the value of the objective function at $(\mathbf{x}, \mathbf{X})$ (i.e. the energy of the possibly fractional labelling $(\mathbf{x}, \mathbf{X})$) for the MAP estimation problem defined over the CRF with parameter $\boldsymbol{\theta}$. Thus the optimal value of the dominating relaxation A is always greater than or equal to the optimal value of relaxation B. We note here that the concept of domination has been used previously in [4] (to compare LP-S with the linear programming relaxation in [11]).

Relaxations A and B are said to be *equivalent* if A dominates B and B dominates A, i.e. their optimal values are equal to each other for all CRFs. A relaxation A is said to *strictly dominate* relaxation B if A dominates B but B does not dominate A. In other words there exists at least one CRF with parameter $\boldsymbol{\theta}$ such that

$$\min_{(\mathbf{x}, \mathbf{X}) \in \mathcal{F}(\text{A})} e(\mathbf{x}, \mathbf{X}; \boldsymbol{\theta}) > \min_{(\mathbf{x}, \mathbf{X}) \in \mathcal{F}(\text{B})} e(\mathbf{x}, \mathbf{X}; \boldsymbol{\theta}). \tag{22}$$

Note that, by definition, the optimal value of any relaxation would always be less than or equal to the energy of the optimal (i.e. the MAP) labelling. Hence, the optimal value of a strictly dominating relaxation A is closer to the optimal value of the MAP estimation IP compared to that of relaxation B. In other words, A provides a tighter lower bound for MAP estimation than B.

**Our Results:**   We prove that LP-S strictly dominates SOCP-MS (see section 3). Further, in section 4, we show that QP-RL is equivalent to SOCP-MS. This implies that LP-S strictly dominates the QP-RL relaxation. In section 5 we generalize the above results by proving that a large class of SOCP (and equivalent QP) relaxations is dominated by LP-S. Based on these results, we propose a novel set of constraints which result in SOCP relaxations that dominate LP-S, QP-RL and SOCP-MS. These relaxations introduce SOC constraints on cycles and cliques formed by the neighbourhood relationship of the CRF. Note that we will only provide the statement of the results here due to page limit. **All the proofs are described in [13]**.

## 3  LP-S vs. SOCP-MS

We now show that for the MAP estimation problem the linear constraints of LP-S are stronger than the SOCP-MS constraints. In other words the feasibility region of LP-S is a strict subset of the feasibility region of SOCP-MS (i.e. $\mathcal{F}(\text{LP-S}) \subset \mathcal{F}(\text{SOCP-MS})$). This in turn would allow us to prove the following theorem.

**Theorem 1:** The LP-S relaxation strictly dominates the SOCP-MS relaxation.

## 4  QP-RL vs. SOCP-MS

We now prove that QP-RL and SOCP-MS are equivalent (i.e. their optimal values are equal for MAP estimation problems defined over all CRFs). Specifically, we consider a vector $\mathbf{x}$ which lies in the

feasibility regions of the QP-RL and SOCP-MS relaxations, i.e. $\mathbf{x} \in [-1,1]^{nh}$. For this vector, we show that the values of the objective functions of the QP-RL and SOCP-MS relaxations are equal. This would imply that if $\mathbf{x}^*$ is an optimal solution of QP-RL for some CRF with parameter $\boldsymbol{\theta}$ then there exists an optimal solution $(\mathbf{x}^*, \mathbf{X}^*)$ of the SOCP-MS relaxation. Further, if $e^Q$ and $e^S$ are the optimal values of the objective functions obtained using the QP-RL and SOCP-MS relaxation, then $e^Q = e^S$.

**Theorem 2:** The QP-RL relaxation and the SOCP-MS relaxation are equivalent.

Theorems 1 and 2 prove that the LP-S relaxation strictly dominates the QP-RL and SOCP-MS relaxations. A natural question that now arises is whether the additive bound of QP-RL (proved in [18]) is applicable to the LP-S and SOCP-MS relaxations. Our next theorem answers this question in an affirmative.

**Theorem 3:** Using the rounding scheme of [18], LP-S and SOCP-MS provide the same additive bound as the QP-RL relaxation, i.e. $\frac{S}{4}$ where $S = \sum_{(a,b) \in \mathcal{E}} \sum_{l_i, l_j \in \mathbf{l}} |\theta^2_{ab;ij}|$ (i.e. the sum of the absolute values of all pairwise potentials). Furthermore, this bound is tight.

The above bound was proved for the case of binary variables (i.e. h = 2) in [8] using a slightly different rounding scheme.

## 5  QP and SOCP Relaxations over Trees and Cycles

We now generalize the results of Theorem 1 by defining a large class of SOCP relaxations which is dominated by LP-S. Specifically, we consider the SOCP relaxations which relax the non-convex constraint $\mathbf{X} = \mathbf{x}\mathbf{x}^\top$ using a set of second order cone (SOC) constraints of the form

$$||(\mathbf{U}^k)^\top \mathbf{x}|| \leq \mathbf{C}^k \bullet \mathbf{X}, k = 1, \cdots, n_C \qquad (23)$$

where $\mathbf{C}^k = \mathbf{U}^k(\mathbf{U}^k)^\top \succeq 0$, for all $k = 1, \cdots, n_C$.

Note that each SOCP relaxation belonging to this class would define an equivalent QP relaxation (similar to the equivalent QP-RL relaxation defined by the SOCP-MS relaxation). Hence, all these QP relaxations will also be dominated by the LP-S relaxation. Before we begin to describe our results in detail, we need to set up some notation as follows.

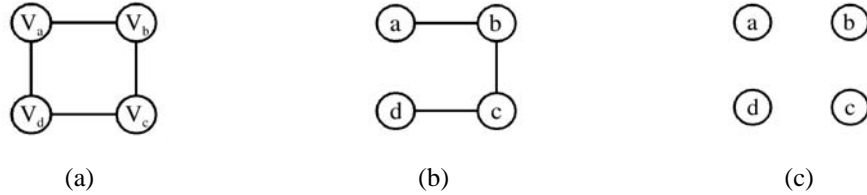

(a)             (b)             (c)

Figure 1: **(a)** *An example* CRF *defined over four variables which form a cycle. Note that the observed nodes are not shown for the sake of clarity of the image.* **(b)** *The set $E^k$ specified by the matrix $\mathbf{C}^k$ shown in equation (25), i.e. $E^k = \{(a,b), (b,c), (c,d)\}$.* **(c)** *The set $V^k = \{a,b,c,d\}$. See text for definitions of these sets.*

**Notation:** We consider an SOC constraint which is of the form described in equation (23), i.e.

$$||(\mathbf{U}^k)^\top \mathbf{x}|| \leq \mathbf{C}^k \bullet \mathbf{X}, \qquad (24)$$

where $k \in \{1, \cdots, n_C\}$. In order to help the reader understand the notation better, we use an example CRF shown in Fig. 1(a). This CRF is defined over four variables $\mathbf{v} = \{v_a, v_b, v_c, v_d\}$ (connected to form a cycle of length 4), each of which take a label from the set $\mathbf{l} = \{l_0, l_1\}$. For this CRF we specify a constraint using a matrix $\mathbf{C}^k \succeq 0$ which is 0 everywhere, except for the following $4 \times 4$ submatrix:

$$\begin{pmatrix} C^k_{aa;00} & C^k_{ab;00} & C^k_{ac;00} & C^k_{ad;00} \\ C^k_{ba;00} & C^k_{bb;00} & C^k_{bc;00} & C^k_{bd;00} \\ C^k_{ca;00} & C^k_{cb;00} & C^k_{cc;00} & C^k_{cd;00} \\ C^k_{da;00} & C^k_{db;00} & C^k_{dc;00} & C^k_{dd;00} \end{pmatrix} = \begin{pmatrix} 2 & 1 & 1 & 0 \\ 1 & 2 & 1 & 1 \\ 1 & 1 & 2 & 1 \\ 0 & 1 & 1 & 2 \end{pmatrix} \qquad (25)$$

Using the SOC constraint shown in equation (24) we define the following two sets: (i) The set $E^k$ is defined such that $(a,b) \in E^k$ if, and only if, it satisfies the following conditions:

$$(a,b) \in \mathcal{E}, \qquad (26)$$

$$\exists l_i, l_j \in \mathbf{l} \text{ such that } C^k_{ab;ij} \neq 0. \tag{27}$$

Recall that $\mathcal{E}$ specifies the neighbourhood relationship for the given CRF. In other words $E^k$ is the subset of the edges in the graphical model of the CRF such that $\mathbf{C}^k$ specifies constraints for the random variables corresponding to those edges. For the example CRF (shown in Fig. 1(a)) and $\mathbf{C}^k$ matrix (in equation (25)), the set $E^k$ obtained is shown in Fig. 1(b). (ii) The set $V^k$ is defined as $a \in V^k$ if, and only if, there exists a $v_b \in \mathbf{v}$ such that $(a, b) \in E^k$. In other words $V^k$ is the subset of hidden nodes in the graphical model of the CRF such that $\mathbf{C}^k$ specifies constraints for the random variables corresponding to those hidden nodes. Fig. 1(c) shows the set $V^k$ for our example SOC constraint.

We also define a weighted graph $G^k = (V^k, E^k)$ whose vertices are specified by the set $V^k$ and whose edges are specified by the set $E^k$. The weight of an edge $(a, b) \in E^k$ is given by $w(a, b)$. Recall that $w(a, b)$ specifies the strength of the pairwise relationship between two neighbouring variables $v_a$ and $v_b$. Thus, for our example SOC constraint, the vertices of this graph are given in Fig. 1(c) while the edges are shown in Fig. 1(b). This graph can be viewed as a subgraph of the graphical model representation for the given CRF.

**Theorem 4:** SOCP relaxations (and the equivalent QP relaxations) which define constraints only using graphs $G^k = (V^k, E^k)$ which form (arbitrarily large) trees are dominated by the LP-S relaxation.

We note that the above theorem can be proved using the results of [24] on *moment constraints* (which imply that LP-S provides the exact solution for the MAP estimation problems defined over tree-structured random fields). However, our alternative proof presented in [13] allows us to generalize the results of Theorem 4 for certain cycles as follows.

**Theorem 5:** When $d(i, j) \geq 0$ for all $l_i, l_j \in \mathbf{l}$, the SOCP relaxations which define constraints only using non-overlapping graphs $G^k$ which form (arbitrarily large) even cycles with all positive or all negative weights are dominated by the LP-S relaxation.

The above theorem can be proved for cycles of any length whose weights are all negative by a similar construction. Further, it also holds true for *odd cycles* (i.e. cycles of odd number of variables) which have only one positive or only one negative weight. However, as will be seen in the next section, unlike trees it is not possible to extend these results for any general cycle.

## 6 Some Useful SOC Constraints

We now describe two SOCP relaxations which include all the marginalization constraints specified in LP-S. Note that the marginalization constraints can be incorporated within the SOCP framework but not in the QP framework.

### 6.1 The SOCP-C Relaxation

The SOCP-C relaxation (where C denotes cycles) defines second order cone (SOC) constraints using positive semidefinite matrices $\mathbf{C}$ such that the graph $G$ (defined in section 5) form cycles. Let the variables corresponding to vertices of one such cycle $G$ of length $c$ be denoted as $\mathbf{v}_C = \{v_b | b \in \{a_1, a_2, \cdots, a_c\}\}$. Further, let $\mathbf{l}_C = \{l_j | j \in \{i_1, i_2, \cdots, i_c\}\} \in \mathbf{l}^c$ be a set of labels for the variables $\mathbf{v}_C$. In addition to the marginalization constraints, the SOCP-C relaxation specifies the following SOC constraint:

$$||\mathbf{U}^\top \mathbf{x}|| \leq \mathbf{C} \bullet \mathbf{X}, \tag{28}$$

such that the graph $G$ defined by the above constraint forms a cycle. The matrix $\mathbf{C}$ is 0 everywhere except the following elements:

$$C_{a_k, a_l, i_k, i_l} = \begin{cases} \lambda_c & \text{if} & k = l, \\ D_c(k, l) & \text{otherwise.} \end{cases} \tag{29}$$

Here $\mathbf{D}_c$ is a $c \times c$ matrix which is defined as follows:

$$D_c(k, l) = \begin{cases} 1 & \text{if} & |k - l| = 1 \\ (-1)^{c-1} & \text{if} & |k - l| = c - 1 \\ 0 & \text{otherwise,} \end{cases} \tag{30}$$

and $\lambda_c$ is the absolute value of the smallest eigenvalue of $\mathbf{D}_c$. In other words the submatrix of $\mathbf{C}$ defined by $\mathbf{v}_C$ and $\mathbf{l}_C$ has diagonal elements equal to $\lambda_c$ and off-diagonal elements equal to the

elements of $\mathbf{D}_c$. Clearly, $\mathbf{C} = \mathbf{U}^\top \mathbf{U} \succeq 0$ since its only non-zero submatrix $\lambda_c I + \mathbf{D}_c$ (where $I$ is a $c \times c$ identity matrix) is positive semifinite. This allows us to define a valid SOC constraint as shown in inequality (28). We choose to define the SOC constraint (28) for only those set of labels $\mathbf{l}_C$ which satisfy the following:

$$\sum_{(a_k,a_l)\in\mathcal{E}} D_c(k,l)\theta^2_{a_k a_l; i_k i_l} \geq \sum_{(a_k,a_l)\in\mathcal{E}} D_c(k,l)\theta^2_{a_k a_l; j_k j_l}, \forall \{j_1, j_2, \cdots, j_c\}. \tag{31}$$

Note that this choice is motivated by the fact that the variables $X_{a_k a_l; i_k i_l}$ corresponding to these sets $\mathbf{v}_C$ and $\mathbf{l}_C$ are assigned trivial values by the LP-S relaxation in the presence of non-submodular terms.

Since marginalization constraints are included in the SOCP-C relaxation, the value of the objective function obtained by solving this relaxation would at least be equal to the value obtained by the LP-S relaxation (i.e. SOCP-C dominates LP-S, see Case II in section 2). We can further show that in the case where $|\mathbf{l}| = 2$ and the constraint (28) is defined over a frustrated cycle (i.e. a cycle with an odd number of non-submodular terms) SOCP-C strictly dominates LP-S. One such example is given in [13]. Note that if the given CRF contains no frustrated cycle, then it can be solved exactly using the method described in [7].

The constraint defined in equation (28) is similar to the (linear) cycle inequality constraints [1] which are given by

$$\sum_{k,l} D_c(k,l)X_{a_k a_l; i_k i_l} \geq 2 - c. \tag{32}$$

We believe that the feasibility region defined by cycle inequalities is a strict subset of the feasibility region defined by equation (28). In other words a relaxation defined by adding cycle inequalities to LP-S would strictly dominate SOCP-C. We are not aware of a formal proof for this. We now describe the SOCP-Q relaxation.

## 6.2   The SOCP-Q Relaxation

In this previous section we saw that LP-S dominates SOCP relaxations whose constraints are defined on trees. However, the SOCP-C relaxation, which defines its constraints using cycles, strictly dominates LP-S. This raises the question whether matrices $\mathbf{C}$, which result in more complicated graphs $G$, would provide an even better relaxation for the MAP estimation problem. In this section, we answer this question in an affirmative. To this end, we define an SOCP relaxation which specifies constraints such that the resulting graph $G$ from a clique. We denote this relaxation by SOCP-Q (where Q indicates cliques).

The SOCP-Q relaxation contains the marginalization constraint and the cycle inequalities (defined above). In addition, it also defines SOC constraints on graphs $G$ which form a clique. We denote the variables corresponding to the vertices of clique $G$ as $\mathbf{v}_Q = \{v_b | b \in \{a_1, a_2, \cdots, a_q\}\}$. Let $\mathbf{l}_Q = \{l_j | j \in \{i_1, i_2, \cdots, i_q\}\}$ be a set of labels for these variables $\mathbf{v}_Q$. Given this set of variables $\mathbf{v}_Q$ and labels $\mathbf{l}_Q$, we define an SOC constraint using a matrix $\mathbf{C}$ of size $nh \times nh$ which is zero everywhere except for the elements $C_{a_k a_l; i_k i_l} = 1$. Clearly, $\mathbf{C}$ is a rank 1 matrix with eigenvalue 1 and eigenvector $\mathbf{u}$ which is zero everywhere except $u_{a_k; i_k} = 1$ where $v_{a_k} \in \mathbf{v}_Q$ and $l_{i_k} \in \mathbf{l}_Q$. This implies that $\mathbf{C} \succeq 0$, which enables us to obtain the following SOC constraint:

$$\left( \sum_k x_{a_k; i_k} \right)^2 \leq q + \sum_{k,l} X_{a_k a_l; i_k i_l}. \tag{33}$$

We choose to specify the above constraint only for the set of labels $\mathbf{l}_Q$ which satisfy the following condition:

$$\sum_{(a_k,a_l)\in\mathcal{E}} \theta^2_{a_k a_l; i_k i_l} \geq \sum_{(a_k,a_l)\in\mathcal{E}} \theta^2_{a_k a_l; j_k j_l}, \forall \{j_1, j_2, \cdots, j_q\}. \tag{34}$$

Again, this choice is motivated by the fact that the variables $X_{a_k a_l; i_k i_l}$ corresponding to these sets $\mathbf{v}_Q$ and $\mathbf{l}_Q$ are assigned trivial values by the LP-S relaxation in the presence of non-submodular pairwise potentials.

When the clique contains a frustrated cycle, it can be shown that SOCP-Q dominates the LP-S relaxation (similar to SOCP-C). Further, using a counter-example, it can proved that the feasibility region given by cycle inequalities is not a subset of the feasibility region defined by constraint (33). One such example is given in [13].

## 7 Discussion

We presented an analysis of approximate algorithms for MAP estimation which are based on convex relaxations. The surprising result of our work is that despite the flexibility in the form of the objective function/constraints offered by QP and SOCP, the LP-S relaxation dominates a large class of QP and SOCP relaxations. It appears that the authors who have previously used SOCP relaxations in the Combinatorial Optimization literature [16] and those who have reported QP relaxation in the Machine Learning literature [18] were unaware of this result. We also proposed two new SOCP relaxations (SOCP-C and SOCP-Q) and presented some examples to prove that they provide a better approximation than LP-S. An interesting direction for future research would be to determine the best SOC constraints for a given MAP estimation problem (e.g. with truncated linear pairwise potentials).

**Acknowledgments:** We thank Pradeep Ravikumar and John Lafferty for careful reading of the manuscript and for pointing out an error in our description of the SOCP-MS relaxation.

## References

[1] F. Barahona and A. Mahjoub. On the cut polytope. *Mathematical Programming*, 36:157–173, 1986.

[2] S. Boyd and L. Vandenberghe. *Convex Optimization*. Cambridge University Press, 2004.

[3] Y. Boykov, O. Veksler, and R. Zabih. Fast approximate energy minimization via graph cuts. *PAMI*, 23(11):1222–1239, 2001.

[4] C. Chekuri, S. Khanna, J. Naor, and L. Zosin. Approximation algorithms for the metric labelling problem via a new linear programming formulation. In *SODA*, 2001.

[5] E. Dalhaus, D. Johnson, C. Papadimitriou, P. Seymour, and M. Yannakakis. The complexity of multi-terminal cuts. *SICOMP*, 23(4):864–894, 1994.

[6] M. Goemans and D. Williamson. Improved approximation algorithms for maximum cut and satisfiability problems using semidefinite programming. *Journal of ACM*, 42:1115–1145, 1995.

[7] P. Hammer, P. Hansen, and B. Simeone. Roof duality, complementation and persistency in quadratic 0-1 optimization. *Mathematical Programming*, 28:121–155, 1984.

[8] P. Hammer and B. Kalantari. A bound on the roof duality gap. Technical Report RRR 46, Rutgers Center for Operations Research, Rutgers University, 1987.

[9] A. Karzanov. Minimum 0-extension of graph metrics. *Euro. J. of Combinatorics*, 19:71–101, 1998.

[10] S. Kim and M. Kojima. Second-order cone programming relaxation of nonconvex quadratic optimization problems. Technical report, Tokyo Institute of Technology, 2000.

[11] J. Kleinberg and E. Tardos. Approximation algorithms for classification problems with pairwise relationships: Metric labeling and Markov random fields. In *STOC*, pages 14–23, 1999.

[12] A. Koster, C. van Hoesel, and A. Kolen. The partial constraint satisfaction problem: Facets and lifting theorems. *Operations Research Letters*, 23(3-5):89–97, 1998.

[13] M. P. Kumar, V. Kolmogorov, and P. H. S. Torr. An analysis of convex relaxations for MAP estimation. Technical report, Oxford Brookes University, 2007. Available at http://cms.brookes.ac.uk/staff/PawanMudigonda/.

[14] M. P. Kumar, P. H. S. Torr, and A. Zisserman. Solving Markov random fields using second order cone programming relaxations. In *CVPR*, volume I, pages 1045–1052, 2006.

[15] J. Lasserre. Global optimization with polynomials and the problem of moments. *SIAM Journal of Optimization*, 11:796–817, 2001.

[16] M. Muramatsu and T. Suzuki. A new second-order cone programming relaxation for max-cut problems. *Journal of Operations Research of Japan*, 43:164–177, 2003.

[17] C. Olsson, A. Eriksson, and F. Kahl. Solving large scale binary quadratic problems: Spectral methods vs. semidefinite programming. In *CVPR*, pages 1–8, 2007.

[18] P. Ravikumar and J. Lafferty. Quadratic programming relaxations for metric labelling and Markov random field MAP estimation. In *ICML*, 2006.

[19] C. Schellewald and C. Schnorr. Subgraph matching with semidefinite programming. In *IWCIA*, 2003.

[20] M. Schlesinger. Sintaksicheskiy analiz dvumernykh zritelnikh singnalov v usloviyakh pomekh (syntactic analysis of two-dimensional visual signals in noisy conditions). *Kibernetika*, 4:113–130, 1976.

[21] R. Szeliski, R. Zabih, D. Scharstein, O. Veksler, V. Kolmogorov, A. Agarwala, M. Tappen, and C. Rother. A comparative study of energy minimization methods for markov random fields. In *ECCV*, pages II: 16–29, 2006.

[22] P. H. S. Torr. Solving Markov random fields using semidefinite programming. In *AISTATS*, 2003.

[23] M. Wainwright, T. Jaakola, and A. Willsky. MAP estimation via agreement on trees: Message passing and linear programming. *IEEE Trans. on Information Theory*, 51(11):3697–3717, 2005.

[24] M. Wainwright and M. Jordan. Graphical models, exponential families, and variational inference. Technical Report 649, University of California, Berkeley, 2003.

[25] M. Wainwright and M. Jordan. Treewidth-based conditions for exactness of the Sherali-Adams and Lasserre relaxations. Technical Report 671, University of California, Berkeley, 2004.

